# Empirical Entropy Manipulation for Real-World Problems

Paul Viola,* Nicol N. Schraudolph, Terrence J. Sejnowski
Computational Neurobiology Laboratory
The Salk Institute for Biological Studies
10010 North Torrey Pines Road
La Jolla, CA 92037-1099
viola@salk.edu

## Abstract

No finite sample is sufficient to determine the density, and therefore the entropy, of a signal directly. Some assumption about either the functional form of the density or about its smoothness is necessary. Both amount to a prior over the space of possible density functions. By far the most common approach is to assume that the density has a parametric form.

By contrast we derive a differential learning rule called EMMA that optimizes entropy by way of kernel density estimation. Entropy and its derivative can then be calculated by sampling from this density estimate. The resulting parameter update rule is surprisingly simple and efficient.

We will show how EMMA can be used to detect and correct corruption in magnetic resonance images (MRI). This application is beyond the scope of existing parametric entropy models.

## 1 Introduction

Information theory is playing an increasing role in unsupervised learning and visual processing. For example, Linsker has used the concept of information maximization to produce theories of development in the visual cortex (Linsker, 1988). Becker and Hinton have used information theory to motivate algorithms for visual processing (Becker and Hinton, 1992). Bell and Sejnowski have used information maximization

to solve the "cocktail party" or signal separation problem (Bell and Sejnowski, 1995). In order to simplify analysis and implementation, each of these techniques makes specific assumptions about the nature of the signals used, typically that the signals are drawn from some parametric density. In practice, such assumptions are very inflexible.

In this paper we will derive a procedure that can effectively estimate and manipulate the entropy of a wide variety of signals using non-parametric densities. Our technique is distinguished by is simplicity, flexibility and efficiency.

We will begin with a discussion of principal components analysis (PCA) as an example of a simple parametric entropy manipulation technique. After pointing out some of PCA's limitation, we will then derive a more powerful non-parametric entropy manipulation procedure. Finally, we will show that the same entropy estimation procedure can be used to tackle a difficult visual processing problem.

## 1.1 Parametric Entropy Estimation

Typically parametric entropy estimation is a two step process. We are given a parametric model for the density of a signal and a sample. First, from the space of possible density functions the most probable is selected. This often requires a search through parameter space. Second, the entropy of the most likely density function is evaluated.

Parametric techniques can work well when the assumed form of the density matches the actual data. Conversely, when the parametric assumption is violated the resulting algorithms are incorrect. The most common assumption, that the data follow the Gaussian density, is especially restrictive. An entropy maximization technique that assumes that data is Gaussian, but operates on data drawn from a non-Gaussian density, may in fact end up minimizing entropy.

## 1.2 Example: Principal Components Analysis

There are a number of signal processing and learning problems that can be formulated as entropy maximization problems. One prominent example is *principal component analysis* (PCA). Given a random variable $X$, a vector $v$ can be used to define a new random variable, $Y_v = X \cdot v$ with variance $\text{Var}(Y_v) = E[(X \cdot v - E[X \cdot v])^2]$. The principal component $\hat{v}$ is the unit vector for which $\text{Var}(Y_{\hat{v}})$ is maximized.

In practice neither the density of $X$ nor $Y_v$ is known. The projection variance is computed from a finite sample, $A$, of points from $X$,

$$\text{Var}(Y_v) \approx \text{Var}_A(Y_v) \equiv E_A[(X \cdot v - E_A[X \cdot v])^2] \ , \tag{1}$$

where $\text{Var}_A(Y_v)$ and $E_A[\cdot]$ are shorthand for the empirical variance and mean evaluated over $A$. Oja has derived an elegant on-line rule for learning $\hat{v}$ when presented with a sample of $X$ (Oja, 1982).

Under the assumption that $X$ is Gaussian is is easily proven that $Y_{\hat{v}}$ has maximum entropy. Moreover, in the absence of noise, $Y_{\hat{v}}$, contains maximal information about $X$. However, when $X$ is *not* Gaussian $Y_{\hat{v}}$ is generally not the most informative projection.

## 2 Estimating Entropy with Parzen Densities

We will now derive a general procedure for manipulating and estimating the entropy of a random variable from a sample. Given a sample of a random variable $X$, we can

construct another random variable $Y = F(X, v)$. The entropy, $h(Y)$, is a function of $v$ and can be manipulated by changing $v$. The following derivation assumes that $Y$ is a vector random variable. The joint entropy of a two random variables, $h(W_1, W_2)$, can be evaluated by constructing the vector random variable, $Y = [W_1, W_2]^T$ and evaluating $h(Y)$.

Rather than assume that the density has a parametric form, whose parameters are selected using maximum likelihood estimation, we will instead use Parzen window density estimation (Duda and Hart, 1973). In the context of entropy estimation, the Parzen density estimate has three significant advantages over maximum likelihood parametric density estimates: (1) it can model the density of any signal provided the density function is smooth; (2) since the Parzen estimate is computed directly from the sample, there is no search for parameters; (3) the derivative of the entropy of the Parzen estimate is simple to compute.

The form of the Parzen estimate constructed from a sample $A$ is

$$P^*(y, A) = \frac{1}{N_A} \sum_{y_A \in A} R(y - y_A) = E_A[R(y - y_A)] \;, \tag{2}$$

where the Parzen estimator is constructed with the window function $R(\cdot)$ which integrates to 1. We will assume that the Parzen window function is a Gaussian density function. This will simplify some analysis, but it is *not* necessary. Any differentiable function could be used. Another good choice is the Cauchy density.

Unfortunately evaluating the entropy integral

$$h(Y) \approx -E[\log P^*(Y, A)] = -\int_{-\infty}^{\infty} \log P^*(y, A) dy$$

is inordinately difficult. This integral can however be approximated as a sample mean:

$$h(Y) \approx h^*(Y) \equiv -E_B[\log P^*(Y, A)] \tag{3}$$

where $E_B[\;]$ is the sample mean taken over the sample $B$. The sample mean converges toward the true expectation at a rate proportional to $1/\sqrt{N_B}$ ($N_B$ is the size of $B$). To reiterate, two samples can be used to estimate the entropy of a density: the first is used to estimate the density, the second is used to estimate the entropy[1]. We call $h^*(Y)$ the EMMA estimate of entropy[2].

One way to extremize entropy is to use the derivative of entropy with respect to $v$. This may be expressed as

$$\frac{d}{dv} h(Y) \approx \frac{d}{dv} h^*(Y) = -\frac{1}{N_B} \sum_{y_B \in B} \frac{\sum_{y_A \in A} \frac{d}{dv} g_\psi(y_B - y_A)}{\sum_{y_A \in A} g_\psi(y_B - y_A)} \tag{4}$$

$$= \frac{1}{N_B} \sum_{y_B \in B} \sum_{y_A \in A} W_y(y_B, y_A) \frac{d}{dv} \frac{1}{2} D_\psi(y_B - y_A) \;, \tag{5}$$

$$\text{where } W_y(y_1, y_2) \equiv \frac{g_\psi(y_1 - y_2)}{\sum_{y_A \in A} g_\psi(y_1 - y_A)} \;, \tag{6}$$

$D_\psi(y) \equiv y^T \psi^{-1} y$, and $g_\psi(y)$ is a multi-dimensional Gaussian with covariance $\psi$. $W_y(y_1, y_2)$ is an indicator of the degree of match between its arguments, in a "soft"

sense. It will approach one if $y_1$ is significantly closer to $y_2$ than any element of $A$. To reduce entropy the parameters $v$ are adjusted such that there is a reduction in the average squared distance between points which $W_y$ indicates are nearby.

## 2.1 Stochastic Maximization Algorithm

Both the calculation of the EMMA entropy estimate and its derivative involve a double summation. As a result the cost of evaluation is quadratic in sample size: $O(N_A N_B)$. While an accurate estimate of empirical entropy could be obtained by using all of the available data (at great cost), a stochastic estimate of the entropy can be obtained by using a random subset of the available data (at quadratically lower cost). This is especially critical in entropy manipulation problems, where the derivative of entropy is evaluated many hundreds or thousands of times. Without the quadratic savings that arise from using smaller samples entropy manipulation would be impossible (see (Viola, 1995) for a discussion of these issues).

## 2.2 Estimating the Covariance

In addition to the learning rate $\lambda$, the covariance matrices of the Parzen window functions, $g_\psi$, are important parameters of EMMA. These parameters may be chosen so that they are optimal in the maximum likelihood sense. For simplicity, we assume that the covariance matrices are diagonal, $\psi = \mathrm{DIAG}(\sigma_1^2, \sigma_2^2, \ldots)$. Following a derivation almost identical to the one described in Section 2 we can derive an equation analogous to (4),

$$\frac{d}{d\sigma_k} h^*(Y) = \frac{1}{N_B} \sum_{y_B \in b} \sum_{y_A \in a} W_y(y_B, y_A) \left(\frac{1}{\sigma_k}\right) \left(\frac{[y]_k^2}{\sigma_k^2} - 1\right) \tag{7}$$

where $[y]_k$ is the $k$th component of the vector $y$. The optimal, or most likely, $\psi$ minimizes $h^*(Y)$. In practice both $v$ and $\psi$ are adjusted simultaneously; for example, while $v$ is adjusted to maximize $h^*(Y_v)$, $\psi$ is adjusted to minimize $h^*(Y_v)$.

# 3 Principal Components Analysis and Information

As a demonstration, we can derive a parameter estimation rule akin to principal components analysis that truly maximizes information. This new EMMA based component analysis (ECA) manipulates the entropy of the random variable $Y_v = X \cdot v$ under the constraint that $|v| = 1$. For any given value of $v$ the entropy of $Y_v$ can be estimated from two samples of $X$ as: $h^*(Y_v) = -E_B[\log E_A[g_\psi(x_B \cdot v - x_A \cdot v)]]$, where $\psi$ is the variance of the Parzen window function. Moreover we can estimate the derivative of entropy:

$$\frac{d}{dv} h^*(Y_v) = \frac{1}{N_B} \sum_B \sum_A W_y(y_B, y_A) \, \psi^{-1}(y_B - y_A)(x_B - x_A) \ ,$$

where $y_A = x_A \cdot v$ and $y_B = x_B \cdot v$. The derivative may be decomposed into parts which can be understood more easily. Ignoring the weighting function $W_y \psi^{-1}$ we are left with the derivative of some unknown function $f(Y_v)$:

$$\frac{d}{dv} f(Y_v) = \frac{1}{N_B N_A} \sum_B \sum_A (y_B - y_A)(x_B - x_A) \tag{8}$$

What then is $f(Y_v)$? The derivative of the squared difference between samples is: $\frac{d}{dv}(y_B - y_A)^2 = 2(y_B - y_A)(x_B - x_A)$ . So we can see that

$$f(Y_v) = \frac{1}{2N_B N_A} \sum_B \sum_A (y_B - y_A)^2$$

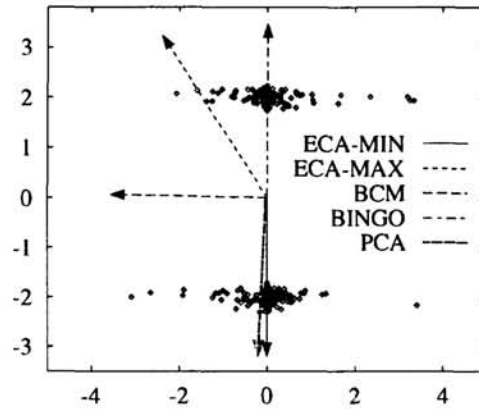

Figure 1: See text for description.

is one half the expectation of the squared difference between pairs of trials of $Y_v$.

Recall that PCA searches for the projection, $Y_v$, that has the largest sample variance. Interestingly, $f(Y_v)$ is precisely the sample variance. Without the weighting term $W_y \psi^{-1}$, ECA would find exactly the same vector that PCA does: the maximum variance projection vector. However because of $W_y$, the derivative of ECA does not act on all points of $A$ and $B$ equally. Pairs of points that are far apart are forced no further apart. Another way of interpreting ECA is as a type of robust variance maximization. Points that might best be interpreted as outliers, because they are very far from the body of other points, play a very small role in the minimization. This robust nature stands in contrast to PCA which is very sensitive to outliers.

For densities that are Gaussian, the maximum entropy projection is the first principal component. In simulations ECA effectively finds the same projection as PCA, and it does so with speeds that are comparable to Oja's rule. ECA can be used both to find the entropy maximizing (ECA-MAX) and minimizing (ECA-MIN) axes. For more complex densities the PCA axis is very different from the entropy maximizing axis. To provide some intuition regarding the behavior of ECA we have run ECA-MAX, ECA-MIN, Oja's rule, and two related procedures, BCM and BINGO, on the same density. BCM is a learning rule that was originally proposed to explain development of receptive fields patterns in visual cortex (Bienenstock, Cooper and Munro, 1982). More recently it has been argued that the rule finds projections that are far from Gaussian (Intrator and Cooper, 1992). Under a limited set of conditions this is equivalent to finding the minimum entropy projection. BINGO was proposed to find axes along which there is a bimodal distribution (Schraudolph and Sejnowski, 1993).

Figure 1 displays a 400 point sample and the projection axes discussed above. The density is a mixture of two clusters. Each cluster has high kurtosis in the horizontal direction. The *oblique* axis projects the data so that it is most uniform and hence has the highest entropy; ECA-MAX finds this axis. Along the *vertical* axis the data is clustered and has low entropy; ECA-MIN finds this axis. The vertical axis also has the highest variance. Contrary to published accounts, the first principal component can in fact correspond to the *minimum* entropy projection. BCM, while it may find minimum entropy projections for some densities, is attracted to the kurtosis along the *horizontal* axis. For this distribution BCM neither minimizes nor maximizes entropy. Finally, BINGO successfully discovers that the *vertical* axis is very bimodal.

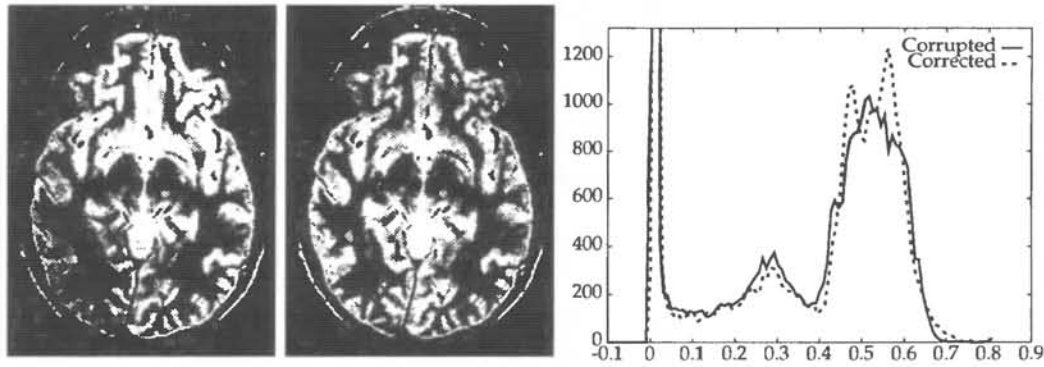

Figure 2: At left: A slice from an MRI scan of a head. Center: The scan after correction. Right: The density of pixel values in the MRI scan before and after correction.

## 4   Applications

EMMA has proven useful in a number of applications. In object recognition EMMA has been used align 3D shape models with video images (Viola and Wells III, 1995). In the area of medical imaging EMMA has been used to register data that arises from differing medical modalities such as magnetic resonance images, computed tomography images, and positron emission tomography (Wells, Viola and Kikinis, 1995).

### 4.1   MRI Processing

In addition, EMMA can be used to process magnetic resonance images (MRI). An MRI is a 2 or 3 dimensional image that records the density of tissues inside the body. In the head, as in other parts of the body, there are a number of distinct tissue classes including: bone, water, white matter, grey matter, and fat. In principle the density of pixel values in an MRI should be clustered, with one cluster for each tissue class. In reality MRI signals are corrupted by a bias field, a multiplicative offset that varies slowly in space. The bias field results from unavoidable variations in magnetic field (see (Wells III et al., 1994) for an overview of this problem).

Because the densities of each tissue type cluster together tightly, an uncorrupted MRI should have relatively low entropy. Corruption from the bias field perturbs the MRI image, increasing the values of some pixels and decreasing others. The bias field acts like noise, adding entropy to the pixel density. We use EMMA to find a low-frequency *correction* field that when applied to the image, makes the pixel density have a lower entropy. The resulting corrected image will have a tighter clustering than the original density.

Call the uncorrupted scan $s(x)$; it is a function of a spatial random variable $x$. The corrupted scan, $c(x) = s(x) + b(x)$ is a sum of the true scan and the bias field. There are physical reasons to believe $b(x)$ is a low order polynomial in the components of $x$. EMMA is used to minimize the entropy of the corrected signal, $h(c(x) - \hat{b}(x, v))$, where $\hat{b}(x, v)$, a third order polynomial with coefficients $v$, is an estimate for the bias corruption.

Figure 2 shows an MRI scan and a histogram of pixel intensity before and after correction. The difference between the two scans is quite subtle: the uncorrected scan is brighter at top right and dimmer at bottom left. This non-homogeneity

makes constructing automatic tissue classifiers difficult. In the histogram of the original scan white and grey matter tissue classes are confounded into a single peak ranging from about 0.4 to 0.6. The histogram of the corrected scan shows much better separation between these two classes. For images like this the correction field takes between 20 and 200 seconds to compute on a Sparc 10.

## 5  Conclusion

We have demonstrated a novel entropy manipulation technique working on problems of significant complexity and practical importance. Because it is based on non-parametric density estimation it is quite flexible, requiring no strong assumptions about the nature of signals. The technique is widely applicable to problems in signal processing, vision and unsupervised learning. The resulting algorithms are computationally efficient.

**Acknowledgements**

This research was support by the Howard Hughes Medical Institute.

## Footnotes

*Author to whom correspondence should be addressed. Current address: M.I.T., 545 Technology Square, Cambridge, MA 02139.

[1]Using a procedure akin to leave-one-out cross-validation a single sample can be used for both purposes.

[2]EMMA is a random but pronounceable subset of the letters in the words "Empirical entropy Manipulation and Analysis".

## References

Becker, S. and Hinton, G. E. (1992). A self-organizing neural network that discovers surfaces in random-dot stereograms. *Nature*, 355:161–163.

Bell, A. J. and Sejnowski, T. J. (1995). An information-maximisation approach to blind separation. In Tesauro, G., Touretzky, D. S., and Leen, T. K., editors, *Advances in Neural Information Processing*, volume 7, Denver 1994. MIT Press, Cambridge.

Bienenstock, E., Cooper, L., and Munro, P. (1982). Theory for the development of neuron selectivity: Orientation specificity and binocular interaction in visual cortex. *Journal of Neuroscience*, 2.

Duda, R. and Hart, P. (1973). *Pattern Classification and Scene Analysis*. Wiley, New York.

Intrator, N. and Cooper, L. N. (1992). Objective function formulation of the bcm theory of visual cortical plasticity: Statistical connections, stability conditions. *Neural Networks*, 5:3–17.

Linsker, R. (1988). Self-organization in a perceptual network. *IEEE Computer*, pages 105–117.

Oja, E. (1982). A simplified neuron model as a principal component analyzer. *Journal of Mathematical Biology*, 15:267–273.

Schraudolph, N. N. and Sejnowski, T. J. (1993). Unsupervised discrimination of clustered data via optimization of binary information gain. In Hanson, S. J., Cowan, J. D., and Giles, C. L., editors, *Advances in Neural Information Processing*, volume 5, pages 499–506, Denver 1992. Morgan Kaufmann, San Mateo.

Viola, P. A. (1995). *Alignment by Maximization of Mutual Information*. PhD thesis, Massachusetts Institute of Technology. MIT AI Laboratory TR 1548.

Viola, P. A. and Wells III, W. M. (1995). Alignment by maximization of mutual information. In *Fifth Intl. Conf. on Computer Vision*, pages 16–23, Cambridge, MA. IEEE.

Wells, W., Viola, P., and Kikinis, R. (1995). Multi-modal volume registration by maximization of mutual information. In *Proceedings of the Second International Symposium on Medical Robotics and Computer Assisted Surgery*, pages 55 – 62. Wiley.

Wells III, W., Grimson, W., Kikinis, R., and Jolesz, F. (1994). Statistical Gain Correction and Segmentation of MRI Data. In *Proceedings of the Computer Society Conference on Computer Vision and Pattern Recognition*, Seattle, Wash. IEEE , *Submitted*.
